# The pigeon as particle filter

**Nathaniel D. Daw**
Center for Neural Science
and Department of Psychology
New York University
daw@cns.nyu.edu

**Aaron C. Courville**
Département d'Informatique
et de recherche opérationnelle
Université de Montréal
aaron.courville@gmail.com

## Abstract

Although theorists have interpreted classical conditioning as a laboratory model of Bayesian belief updating, a recent reanalysis showed that the key features that theoretical models capture about learning are artifacts of averaging over subjects. Rather than learning smoothly to asymptote (reflecting, according to Bayesian models, the gradual tradeoff from prior to posterior as data accumulate), subjects learn suddenly and their predictions fluctuate perpetually. We suggest that abrupt and unstable learning can be modeled by assuming subjects are conducting inference using sequential Monte Carlo sampling with a small number of samples — one, in our simulations. Ensemble behavior resembles exact Bayesian models since, as in particle filters, it averages over many samples. Further, the model is capable of exhibiting sophisticated behaviors like retrospective revaluation at the ensemble level, even given minimally sophisticated individuals that do not track uncertainty in their beliefs over trials.

## 1 Introduction

A central tenet of the Bayesian program is the representation of beliefs by distributions, which assign probability to each of a set of hypotheses. The prominent theoretical status accorded to such ambiguity seems rather puzzlingly at odds with the all-or-nothing nature of our everyday perceptual lives. For instance, subjects observing ambiguous or rivalrous visual displays famously report experiencing either percept alternately and exclusively; for even the most fervent Bayesian, it seems impossible simultaneously to interpret the Necker cube as potentially facing either direction [1].

A longstanding laboratory model for the formation of beliefs and their update in light of experience is Pavlovian conditioning in animals, and analogously structured prediction tasks in humans. There is a rich program of reinterpreting data from such experiments in terms of statistical inference [2, 3, 4, 5, 6]. The data do appear in a number of respects to reflect key features of the Bayesian ideal — specifically, that subjects represent beliefs as distributions with uncertainty and appropriately employ it in updating them in light of new evidence. Most notable in this respect are *retrospective revaluation* phenomena (e.g., [7]), which demonstrate that subjects are able to revise previously favored beliefs in a way suggesting that they had entertained alternative hypotheses all along [6].

However, the data addressed by such models are, in almost all cases, averages over large numbers of subjects. This raises the question whether individuals really exhibit the sophistication attributed to them, or if it instead somehow emerges from the ensemble. Recent work by Gallistel and colleagues [8] frames the problem particularly sharply. Whereas subject-averaged responses exhibit smooth learning curves approaching asymptote (interpreted by Bayesian modelers as reflecting the gradual tradeoff from prior to posterior as data accumulate), individual records exhibit neither smooth learning nor steady asymptote. Instead responding emerges abruptly and fluctuates perpetually. These analyses soundly refute all previous quantitative theories of learning in these tasks: both Bayesian and traditional associative learning.

Here we suggest that individuals' behavior in conditioning might be understood in terms of Monte Carlo methods for sequentially *sampling* different hypotheses (e.g., [9]). Such a model preserves the insights of a statistical framing while accounting for the characteristics of individual records. Through the metaphor of particle filtering, it also explains why exact Bayesian reasoning is a good account of the ensemble. Finally, it addresses another common criticism of Bayesian models: that they attribute wildly intractable computations to the individual. A similar framework has also recently been used to characterize human categorization learning [10].

To make our point in the most extreme way, and to explore the most novel corner of the model space, we here develop as proof of concept the idea that (as with percepts in the Necker cube) subjects sample only a single hypothesis at a time. That is, we treat them as particle filters employing only one particle. We show that even given individuals of such minimal capacity, sophisticated effects like retrospective revaluation can emerge in the ensemble. Clearly intermediate models are possible, either employing more samples or mixtures of sampling and exact methods within the individual, and the insights developed here will extend to those cases. We therefore do not mean to defend the extreme claim that subjects never track or employ uncertainty — we think this would be highly maladaptive — but instead intend to explore the role of sampling and also point out how poor is the evidentiary record supporting more sophisticated accounts, and how great is the need for better experimental and analytical methods to test them.

## 2    Model

### 2.1    Conditioning as exact filtering

In conditioning experiments, a subject (say, a dog) experiences outcomes ("reinforcers," say, food) paired with stimuli (say, a bell). That subjects learn thereby to predict outcomes on the basis of antecedent stimuli is demonstrated by the finding that they emit anticipatory behaviors (such as salivation to the bell) which are taken directly to reflect the expectation of the outcome. Human experiments are analogously structured, but using various cover stories (such as disease diagnosis) and with subjects typically simply asked to state their beliefs about how much they expect the outcome.

A standard statistical framing for such a problem [5], which we will adopt here, is to assume that subjects are trying to learn the conditional probability $P(r \mid \mathbf{x})$ of (real-valued) outcomes $r$ given (vector-valued) stimuli $\mathbf{x}$. One simple generative model is to assume that each stimulus $x_i$ (bells, lights, tones) produces reinforcement according to some unknown parameter $w_i$; that the contributions of multiple stimuli sum; and that the actual reward is Gaussian in the the aggregate. That is, $P(r \mid \mathbf{x}) = \mathcal{N}(\mathbf{x} \cdot \mathbf{w}, \sigma_o^2)$, where we take the variance parameter as known. The goal of the subject is then to infer the unknown weights in order to predict reinforcement. If we further assume the weights $\mathbf{w}$ can change with time, and take that change as Gaussian diffusion,

$$P(\mathbf{w}_{t+1} \mid \mathbf{w}_t) = \mathcal{N}(\mathbf{w}_t, \sigma_d^2 \mathbf{I}) \tag{1}$$

then we complete the well known generative model for which Bayesian inference about the weights can be accomplished using the Kalman filter algorithm [5]. Given a Gaussian prior on $\mathbf{w}_0$, the posterior distribution $P(\mathbf{w_t} \mid \mathbf{x}_{1..t}, r_{1...t})$ also takes a Gaussian form, $\mathcal{N}(\hat{\mathbf{w}}_t, \Sigma_t)$, with the mean and covariance given by the recursive Kalman filter update equations.

Returning to conditioning, a subject's anticipatory responding to test stimulus $\mathbf{x}_t$ is taken to be proportional to her expectation about $r_t$ conditional on $\mathbf{x}_t$, marginalizing out uncertainty over the weights. $E(r_t \mid \mathbf{x}_t, \hat{\mathbf{w}}_t, \Sigma_t) = \mathbf{x}_t \cdot \hat{\mathbf{w}}_t$.

### 2.2    Conditioning as particle filtering

Here we assume instead that subjects do not maintain uncertainty in their posterior beliefs, via covariance $\Sigma_t$, but instead that subject $L$ maintains a point estimate $\widetilde{\mathbf{w}}_t^L$ and treats it as true with certainty. Even given such certainty, because of diffusion intervening between $t$ and $t + 1$, $\widetilde{\mathbf{w}}_{t+1}^L$ will be uncertain; let us assume that she recursively samples her new point estimate $\widetilde{\mathbf{w}}_{t+1}^L$ from the posterior given this diffusion and the new observation $\mathbf{x}_{t+1}, r_{t+1}$:

$$\widetilde{\mathbf{w}}_{t+1}^L \sim P(\mathbf{w}_{t+1}^L \mid \mathbf{w}_t = \widetilde{\mathbf{w}}_t^L, \mathbf{x}_{t+1}, r_{t+1}) \tag{2}$$

This is simply a Gaussian given by the standard Kalman filter equations. In particular, the mean of the sampling distribution is $\widetilde{\mathbf{w}}_t^L + \mathbf{x}_{t+1}\kappa(r_{t+1} - \mathbf{x}_{t+1} \cdot \widetilde{\mathbf{w}}_t)$. Here the Kalman gain $\kappa = \sigma_d^2/(\sigma_d^2 + \sigma_o^2)$ is constant; the *expected* update in $\widetilde{\mathbf{w}}$, then, is just that given by the Rescorla-Wagner [11] model.

Such seemingly peculiar behavior may be motivated by the observation that, assuming that the initial $\widetilde{\mathbf{w}}_0^L$ is sampled according to the prior, this process also describes the evolution of a single sample in particle filtering by sequential importance sampling, with Equation 2 as the optimal proposal distribution [9]. (In this algorithm, particles evolve independently by sequential sampling, and do not interact except for resampling.)

Of course, the idea of such sampling algorithms is that one can estimate the true posterior over $\mathbf{w}_t$ by averaging over particles. In importance sampling, the average must be weighted according to *importance weights*. These (here, the product of $P(r_{t+1} \mid \mathbf{x}_{t+1}, \mathbf{w}_t = \widetilde{\mathbf{w}}_t^L)$ over each $t$) serve to squelch the contribution of particles whose trajectories turn out to be conditionally more unlikely given subsequent observations. If subjects were to behave in accord with this model, then this would give us some insight into the ensemble average behavior, though if computed without importance reweighting, the ensemble average will appear to learn more slowly than the true posterior.

## 2.3 Resampling and jumps

One reason why subjects might employ sampling is that, in generative models more interesting than the toy linear, Gaussian one used here, Bayesian reasoning is notoriously intractable. However, the approximation from a small number of samples (or in the extreme case considered here, one sample) would be noisy and poor. As we can see by comparing the particle filter update rule of Equation 2 to the Kalman filter, because the subject-as-single-sample does not carry uncertainty from trial to trial, she is systematically overconfident in her beliefs and therefore tends to be more reluctant than optimal in updating them in light of new evidence (that is, the Kalman gain is low). This is the individual counterpart to the slowness at the ensemble level, and at the ensemble level, it can be compensated for by importance reweighting and also by *resampling* (for instance, standard sequential importance resampling; [12, 9]). Resampling kills off conditionally unlikely particles and keeps most samples in conditionally likely parts of the space, with similar and high importance weights. Since optimal reweighting and resampling both involve normalizing importance weights over the ensemble, they are not available to our subject-as-sample.

However, there are some generative models that are more forgiving of these problems. In particular, consider Yu and Dayan's [13] diffusion-jump model, which replaces Equation 1 with

$$P(\mathbf{w}_{t+1} \mid \mathbf{w}_t) = (1 - \pi)\mathcal{N}(\mathbf{w}_t, \sigma_d^2\mathbf{I}) + \pi\mathcal{N}(0, \sigma_j^2\mathbf{I}) \tag{3}$$

with $\sigma_j \gg \sigma_d$. Here, the weights usually diffuse as before, but occasionally (with probability $\pi$) are regenerated anew. (We refer to these events as "jumps" and the previous model of Equation 1 as a "no-jump" model, even though, strictly speaking, diffusion is accomplished by smaller jumps.) Since optimal inference in this model is intractable (the number of modes in the posterior grows exponentially) Yu and Dayan [13] propose maintaining a simplified posterior: they make a sort of maximum likelihood determination whether a jump occurred or not; conditional on this the posterior is again Gaussian and inference proceeds as in the Kalman filter.

If we use Equation 3 together with the one-sample particle filtering scheme of Equation 2, then we simplify the posterior still further by not carrying over uncertainty from trial to trial, but instead only a point estimate. As before, at each step, we sample from the posterior $P(\mathbf{w}_{t+1}^L \mid \mathbf{w}_t = \widetilde{\mathbf{w}}_t^L, \mathbf{x}_{t+1}, r_{t+1})$ given total confidence in our previous estimate. This distribution now has two modes, one representing the posterior given that a jump occurred, the other representing the posterior given no jump.

Importantly, we are more likely to infer a jump, and resample from scratch, if the observation $r_{t+1}$ is far from that expected under the hypothesis of no jump, $\mathbf{x}_{t+1} \cdot \widetilde{\mathbf{w}}_t^L$. Specifically, the probability that no jump occurred (and that we therefore resample according to the posterior distribution given drift — effectively, the chance that the sample "survives" as it would have in the no-jump Kalman filter) — is proportional to $P(r_{t+1} \mid \mathbf{x}_{t+1}, \mathbf{w_t} = \widetilde{\mathbf{w}}_t^L, \text{no jump})$. This is also the factor that the trial would contribute to the importance weight in the no-jump Kalman filter model of the previous section. The importance weight, in turn, is also the factor that would determine the chance that a particle would be selected during an exact resampling step [12, 9].

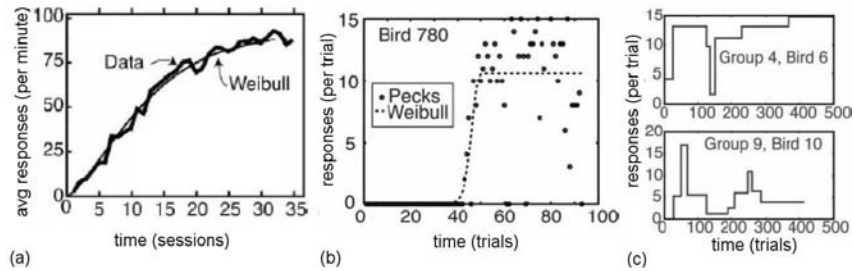

Figure 1: Aggregate versus individual behavior in conditioning, figures adapted with permission from [8], copyright 2004 by The National Academy of Sciences of the USA. (a) Mean over subjects reveals smooth, slow acquisition curve (timebase is in sessions). (b) Individual records are noisier and with more abrupt changes (timebase is in trials). (c) Examples of fits to individual records assuming the behavior is piecewise Poisson with abrupt rate shifts.

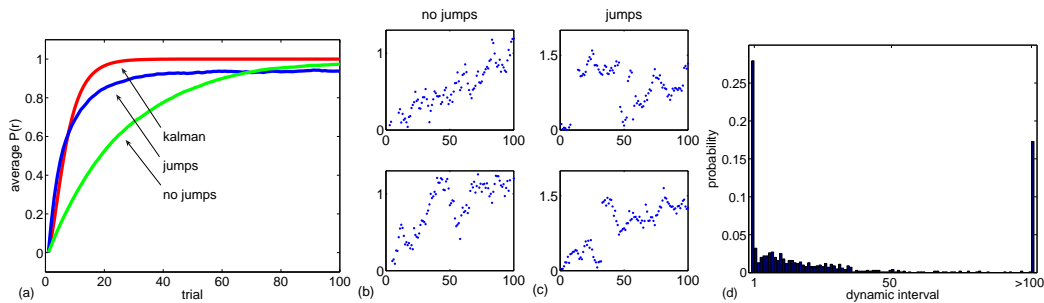

Figure 2: Simple acquisition in conditioning, simulations using particle filter models. (a) Mean behavior over samples for jump ($\pi = 0.075$; $\sigma_j = 1$; $\sigma_d = 0.1$; $\sigma_o = 0.5$) and no-jump ($\pi = 0$) particle filter models of conditioning, plotted against exact Kalman filter for same parameters (and $\pi = 0$). (b) Two examples of individual subject traces for the no-jump particle filter model. (c) Two examples of individual subject traces for the particle filter model incorporating jumps. (d) Distribution over individuals using the jump model of the "dynamic interval" of acquisition, that is the number of trials over which responding grows from negligible to near-asymptotic levels.

There is therefore an analogy between sampling in this model and sampling with resampling in the simpler generative model of Equation 1. Of course, this cannot exactly accomplish optimal resampling, both because the chance that a particle survives should be normalized with respect to the population, and because the distribution from which a non-surviving particle resamples should also depend on the ensemble distribution. However, it has a similar qualitative effect of suppressing conditionally unlikely samples and replacing them ultimately with conditionally more likely ones.

We can therefore view the jumps of Equation 3 in two ways. First, they could correctly model a jumpy world; by periodically resetting itself, such a world would be relatively forgiving of the tendency for particles in sequential importance sampling to turn out conditionally unlikely. Alternatively, the jumps can be viewed as a fiction effectively encouraging a sort of resampling to improve the performance of low-sample particle filtering in the *non*-jumpy world of Equation 1. Whatever their interpretation, as we will show, they are critical to explaining subject behavior in conditioning.

## 3 Acquisition

In this and the following section, we illustrate the behavior of individuals and of the ensemble in some simple conditioning tasks, comparing particle filter models with and without jumps (Equations 1 and 3).

Figure 1 reproduces some data reanalyzed by Gallistel and colleagues [8], who quantify across a number of experiments what had long been anecdotally known about conditioning: that individual

records look nothing like the averages over subjects that have been the focus of much theorizing. Consider the simplest possible experiment, in which a stimulus $A$ is paired repeatedly with food. (We write this as $A+$.) Averaged learning curves slowly and smoothly climb toward asymptote (Figure 1a, here the anticipatory behavior measured is pigeons pecking), just as does the estimate of the mean, $\hat{w}_A$, in the Kalman filter models.

Viewed in individual records (Figure 1b), the onset of responding is much more abrupt (often it occurred in a single trial), and the subsequent behavior much more variable. The apparently slow learning results from the average over abrupt transitions occurring at a range of latencies. Gallistel et al. [8] characterized the behavior as piecewise Poisson with instantaneous rate changes (Figure 1c). These results present a challenge to the bulk of models of conditioning — not just Bayesian ones, but also associative learning theories like the seminal model of Rescorla & Wagner [11] ubiquitously produce smooth, asymptoting learning curves of a sort that these data reveal to be essentially an artifact of averaging.

One further anomaly with Bayesian models even as accounts for the average curves is that acquisition is absurdly slow from a normative perspective — it emerges long after subjects using reasonable priors would be highly certain to expect reward. This was pointed out by Kakade and Dayan [5], who also suggested an account for why the slow acquisition might actually be normative due to unaccounted priors caused by pretraining procedures known as hopper training. However, Balsam and colleagues later found that manipulating the hopper pretraining did not speed learning [14].

Figure 2 illustrates individual and group behavior for the two particle filter models. As expected, at the ensemble level (Figure 2a), particle filtering without jumps learns slowly, when averaged without importance weighting or resampling and compared to the optimal Kalman filter for the same parameters. As shown, the inclusion of jumps can speed this up.

In individual traces using the jumps model (Figure 2c) frequent sampled jumps both at and after acquisition of responding capture the key qualitative features of the individual records: the abrupt onset and ongoing instability. The inclusion of jumps in the generative model is key to this account: as shown in Figure 2b, without these, behavior changes more smoothly. In the jump model, when a jump is sampled, the posterior distribution conditional on the jump having occurred is centered near the observed $r_t$, meaning that the sampled weight will most likely arrive immediately near its asymptotic level. Figure 2d shows that such an abrupt onset of responding is the modal behavior of individuals. Here (after [8]), we have fit each individual run from the jump-model simulations with a sigmoidal Weibull function, and defined the "dynamic interval" over which acquisition occurs as the number of trials during which this fit function rises from 10% to 90% of its asymptotic level. Of course, the monotonic Weibull curve is not a great characterization of the individual's noisy predictions, and this mismatch accounts for the long tail of the distribution. Nevertheless, the cumulative distribution from our simulations closely matches the proportions of animals reported as achieving various dynamic intervals when the same analysis was performed on the pigeon data [8].

These simulations demonstrate, first, how sequential sampling using a very low number of samples is a good model of the puzzling features of individual behavior in acquisition, and at the same time clarify why subject-averaged records resemble the results of exact inference. Depending on the presumed frequency of jumps (which help to compensate for this problem) the fact that these averages are of course computed without importance weighting may also help to explain the apparent slowness of acquisition. This could be true regardless of whether other factors, such as those posited by Kakade and Dayan [5], also contribute.

## 4    Retrospective revaluation

So far, we have shown that sequential sampling provides a good qualitative characterization of individual behavior in the simplest conditioning experiments. But the best support for sophisticated Bayesian models of learning comes from more demanding tasks such as retrospective revaluation. These tasks give the best indication that subjects maintain something more than a point estimate of the weights, and instead strongly suggest that they maintain a full joint distribution over them. However, as we will show here, this effect can actually emerge due to covariance information being implicitly represented in the ensemble of beliefs over subjects, even if all the individuals are one-particle samplers.

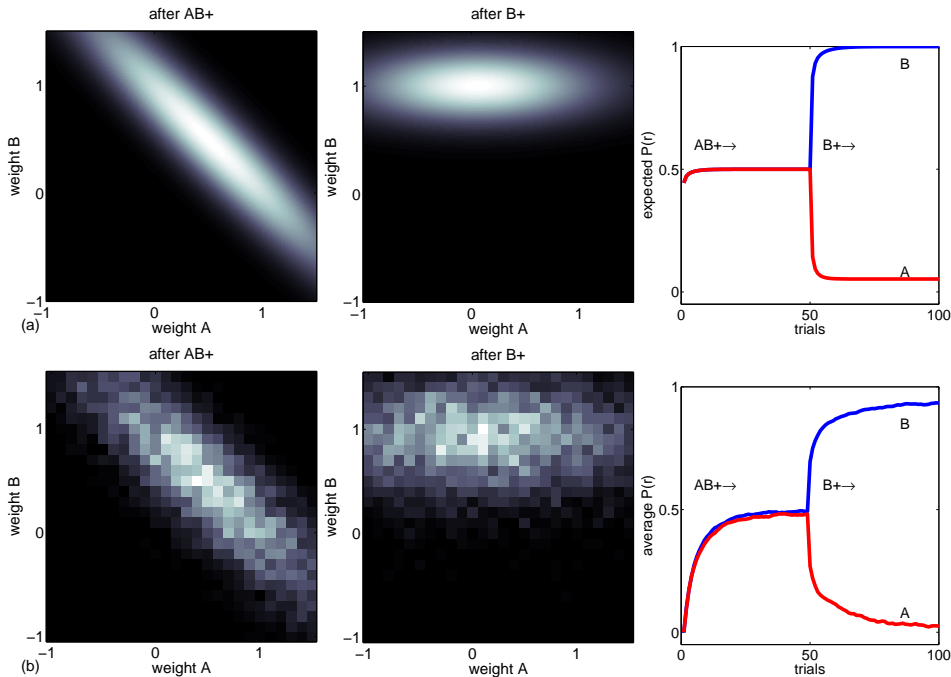

Figure 3: Simulations of backward blocking effect, using exact Kalman filter (a) and particle filter model with jumps (b). Left, middle: Joint distributions over $w_A$ and $w_B$ following first-phase $AB+$ training (left) and second phase $B+$ training (middle). For the particle filter, these are derived from the histogram of individual particles' joint point beliefs about the weights. Right: Mean beliefs about $w_A$ and $w_B$, showing development of backward blocking. Parameters as in Figure 2.

Retrospective revaluation refers to how the interpretation of previous experience can be changed by subsequent experience. A typical task, called *backward blocking* [7], has two phases. First, two stimuli, $A$ and $B$, are paired with each other and reward ($AB+$), so that both develop a moderate level of responding. In the second phase, $B$ alone is paired with reward ($B+$), and then the prediction to $A$ alone is probed. The typical finding is that responding to $A$ is attenuated; the intuition is that the $B+$ trials suggested that $B$ alone was responsible for the reward received in the $AB+$ trials, so the association of $A$ with reward is retrospectively discounted. Such retrospective revaluation phenomena are hard to demonstrate in animals (though see [15]) but robust in humans [7].

Kakade and Dayan [6] gave a more formal analysis of the task in terms of the Kalman filter model. In particular they point out that conditonal on the initial $AB+$ trials, the model will infer an *anti-correlated* joint distribution over $w_A$ and $w_B$ — i.e., that they together add up to about one. This is represented in the covariance $\Sigma$; the joint distribution is illustrated in Figure 3a (left). Subsequent $B+$ training indicates that $w_B$ is high, which means, given its posterior anticorrelation with $w_A$, that the latter is likely low. Note that this explanation seems to turn crucially on the representation of the full joint distribution over the weights, rather than just a point estimate.

Contrary to this intuition, Figure 3b demonstrates the same thing in the particle filter model with jumps. At the end of $AB+$ training, the subjects *as an ensemble* represent the anti-correlated joint distribution over the weights, even though each individual maintains only a particular point belief. Moreover, $B+$ training causes an aggregate backward blocking effect. This is because individuals who believe that $w_A$ is high tend also to believe that $w_B$ is low, which makes them most likely to sample that a jump has occurred during subsequent $B+$ training. The samples most likely to stay in place already have $\widetilde{w}_A$ low and $\widetilde{w}_B$ high; beliefs about $w_A$ are, on average, thereby reduced, producing the backward blocking effect in the ensemble.

Note that this effect depends on the subjects sampling using a generative model that admits of jumps (Equation 3). Although the population implicitly represents the posterior covariance between $w_A$ and $w_B$ even using the diffusion model with no jumps (Equation 1; simulations not illustrated), sub-

sequent $B+$ training has no tendency to suppress the relevant part of the posterior, and no backward blocking effect is seen. Again, this traces to the lack of a mechanism for downweighting samples that turn out to be conditionally unlikely.

## 5 Discussion

We have suggested that individual subjects in conditioning experiments behave as though they are sequentially sampling hypotheses about the underlying weights: like particle filters using a single sample. This model reproduces key and hitherto theoretically troubling features of individual records, and also, rather more surprisingly, has the ability to reproduce more sophisticated behaviors that had previously been thought to demonstrate that subjects represented distributions in a fully Bayesian fashion. One practical problem with particle filtering using a single sample is the lack of distributional information to allow resampling or reweighting; we have shown that use of a particular generative model previously proposed by Yu and Dayan [13] (involving sudden shocks that effectively accomplish resampling) helps to compensate qualitatively if not quantitatively for this failing. This mechanism is key to all of our results.

The present work echoes and formalizes a long history of ideas in psychology about hypothesis testing and sudden insight in learning, going back to Thorndike's puzzle boxes. It also complements a recent model of human categorization learning [10], which used particle filters to sample (sparsely or even with a single sample) over possible clusterings of stimuli. That work concentrated on trial ordering effects arising from the sparsely represented posterior (see also [16]); here we concentrate on a different set of phenomena related to individual versus ensemble behavior.

Gallistel and colleagues' [8] demonstration that individual learning curves exhibit none of the features of the ensemble average curves that had previously been modeled poses rather a serious challenge for theorists: After all, what does it mean to model only the ensemble? Surely the individual subject is the appropriate focus of theory — particularly given the evolutionary rationale often advanced for Bayesian modeling, that individuals who behave rationally will have higher fitness. The present work aims to refocus theorizing on the individual, while at the same time clarifying why the ensemble may be of interest. (At the group level, there may also be a fitness advantage to spreading different beliefs — say, about productive foraging locations — across subjects rather than having the entire population gravitate toward the "best" belief. This is similar to the phenomenon of mixed strategy equilibrium in multiplayer games, and may provide an additional motivation for sampling.)

Previous models fail to predict any intersubject variability because they incorporate no variation in either the subjects' beliefs or in their responses given their beliefs. We have suggested that the structure in response timeseries suggests a prominent role for intersubject variability in the beliefs, due to sampling. There is surely also noise in the responding, which we do not model, but for this alone to rescue previous models, one would have to devise some other explanation for the noise's structure. (For instance, if learning is monotonic, simple IID output noise would not predict sustained excursions away from asymptote as in Fig 1c.) Similarly, nonlinearity in the performance function relating beliefs to response rates might help to account for the sudden onset of responding even if learning is smooth, but would not address the other features of the data.

In addition to addressing the empirical problem of fit to the individual, sampling also answers an additional problem with Bayesian models: that they attribute to subjects the capacity for radically intractable calculations. While the simple Kalman filter used here is tractable, there has been a trend in modeling human and animal learning toward assuming subjects perform inference about model structure (e.g., recovering structural variables describing how different latent causes interact to produce observations; [4, 3, 2]). Such inference cannot be accomplished exactly using simple recursive filtering like the Kalman filter. Indeed, it is hard to imagine any approach other than sequentially sampling one or a small number of hypothetical model structures, since even with the structure known, there typically remains a difficult parametric inference problem. The present modeling is therefore motivated, in part, toward this setting.

While in our model, subjects do not explicitly carry uncertainty about their beliefs from trial to trial, they do maintain hyperparameters (controlling the speed of diffusion, the noise of observations, and the probability of jumps) that serve as a sort of constant proxy for uncertainty. We might expect them

to adjust these so as to achieve the best performance; because the inference is anyway approximate, the veridical, generative settings of these parameters will not necessarily perform the best.

Of course, the present model is only the simplest possible sketch, and there is much work to do in developing it. In particular, it would be useful to develop less extreme models in which subjects either rely on sampling with more particles, or on some combination of sampling and exact inference. We posit that many of the insights developed here will extend to such models, which seem more realistic since *exclusive* use of low-sample particle filtering would be extremely brittle and unreliable. (The example of the Necker cube also invites consideration of Markov Chain Monte Carlo sampling for exploration of multimodal posteriors even in nonsequential inference [1] — such methods are clearly complementary.) However, there is very little information available about individual-level behavior to constrain the details of approximate inference. The present results on backward blocking stress again the perils of averaging and suggest that data must be analyzed much more delicately if they are ever to bear on issues of distributions and uncertainty. In the case of backward blocking, if our account is correct, there should be a correlation, over individuals, between the degree to which they initially exhibited a low $\widetilde{w}_B$ and the degree to which they subsequently exhibited a backward blocking effect. This would be straightforward to test. More generally, there has been a recent trend [17] toward comparing models against raw trial-by-trial data sets according to the cumulative log-likelihood of the data. Although this measure aggregates over trials and subjects, it measures the average goodness of fit, not the goodness of fit to the average, making it much more sensitive for purposes of studying the issues discussed in this article.

# References

[1] P Schrater and R Sundareswara. Theory and dynamics of perceptual bistability. In *NIPS 19*, 2006.

[2] TL Griffiths and JB Tenenbaum. Structure and strength in causal induction. *Cognit Psychol*, 51:334–384, 2005.

[3] AC Courville, ND Daw, and DS Touretzky. Similarity and discrimination in classical conditioning: A latent variable account. In *NIPS 17*, 2004.

[4] AC Courville, ND Daw, GJ Gordon, and DS Touretzky. Model uncertainty in classical conditioning. In *NIPS 16*, 2003.

[5] S Kakade and P Dayan. Acquisition and extinction in autoshaping. *Psychol Rev*, 109:533–544, 2002.

[6] S Kakade and P Dayan. Explaining away in weight space. In *NIPS 13*, 2001.

[7] DR Shanks. Forward and backward blocking in human contingency judgement. *Q J Exp Psychol B*, 37:1–21, 1985.

[8] CR Gallistel, S Fairhurst, and P Balsam. The learning curve: Implications of a quantitative analysis. *Proc Natl Acad Sci USA*, 101:13124–13131, 2004.

[9] A Doucet, S Godsill, and C Andrieu. On sequential Monte Carlo sampling methods for Bayesian filtering. *Stat Comput*, 10:197–208, 2000.

[10] AN Sanborn, TL Griffiths, and DJ Navarro. A more rational model of categorization. In *CogSci 28*, 2006.

[11] RA Rescorla and AR Wagner. A theory of Pavlovian conditioning: The effectiveness of reinforcement and non-reinforcement. In AH Black and WF Prokasy, editors, *Classical Conditioning, 2: Current Research and Theory*, pages 64–69. 1972.

[12] DB Rubin. Using the SIR algorithm to simulate posterior distributions. In JM Bernardo, MH DeGroot, DV Lindley, and AFM Smith, editors, *Bayesian Statistics, Vol. 3*, pages 395–402. 1988.

[13] AJ Yu and P Dayan. Expected and unexpected uncertainty: ACh and NE in the neocortex. In *NIPS 15*, 2003.

[14] PD Balsam, S Fairhurst, and CR Gallistel. Pavlovian Contingencies and Temporal Information. *J Exp Psychol Anim Behav Process*, 32:284–295, 2006.

[15] RR Miller and H Matute. Biological significance in forward and backward blocking: Resolution of a discrepancy between animal conditioning and human causal judgment. *J Exp Psychol Gen*, 125:370–386, 1996.

[16] ND Daw, AC Courville, and P Dayan. Semi-rational models of cognition: The case of trial order. In N Chater and M Oaksford, editors, *The Probabilistic Mind*. 2008. (in press).

[17] ND Daw and K Doya. The computational neurobiology of learning and reward. *Curr Opin Neurobiol*, 16:199–204, 2006.

